# Playing Pinball with non-invasive BCI

**Michael W. Tangermann**
Machine Learning Laboratory
Berlin Institute of Technology
Berlin, Germany
schroedm@cs.tu-berlin.de

**Matthias Krauledat**
Machine Learning Laboratory
Berlin Institute of Technology
Berlin, Germany
kraulem@cs.tu-berlin.de

**Konrad Grzeska**
Machine Learning Laboratory
Berlin Institute of Technology
Berlin, Germany
konradg@cs.tu-berlin.de

**Max Sagebaum**
Machine Learning Laboratory
Berlin Institute of Technology
Berlin, Germany
max.sagebaum@first.fraunhofer.de

**Carmen Vidaurre**
Machine Learning Laboratory
Berlin Institute of Technology
Berlin, Germany
vidcar@cs.tu-berlin.de

**Benjamin Blankertz**
Machine Learning Laboratory
Berlin Institute of Technology
Berlin, Germany
blanker@cs.tu-berlin.de

**Klaus-Robert Müller**
Machine Learning Laboratory, Berlin Institute of Technology, Berlin, Germany
krm@cs.tu-berlin.de

## Abstract

Compared to invasive Brain-Computer Interfaces (BCI), non-invasive BCI systems based on Electroencephalogram (EEG) signals have not been applied successfully for precisely timed control tasks. In the present study, however, we demonstrate and report on the interaction of subjects with a real device: a pinball machine. Results of this study clearly show that *fast and well-timed* control well beyond chance level is possible, even though the environment is extremely rich and requires precisely timed and complex predictive behavior. Using machine learning methods for mental state decoding, BCI-based pinball control is possible within the first session without the necessity to employ lengthy subject training. The current study shows clearly that very compelling control with excellent timing and dynamics is possible for a non-invasive BCI.

## 1 Introduction

Brain computer interfaces (BCI) have seen a rapid development towards faster and more user-friendly systems for thought-based control of devices such as video games, wheel chairs, robotic devices etc. While a full control of even complex trajectories has become possible for invasive BCIs [1, 2, 3], non-invasive EEG-based systems have been considered hardly able to provide such high information transfer rates between man and machine [4, 5].

This paper will show evidence that real-time BCI control of a machine is possible with little subject training. The machine studied (a standard pinball machine, see Fig. 1 requires only two classes for control but a very fast and precise reaction; predictive behavior and learning are mandatory. We

consider it a formidable platform for studying timing and dynamics of brain control in real-time interaction with a physical machine. Furthermore this paradigm is well suited for future investigations of mental states during complex real-time tasks and decision-making processes.

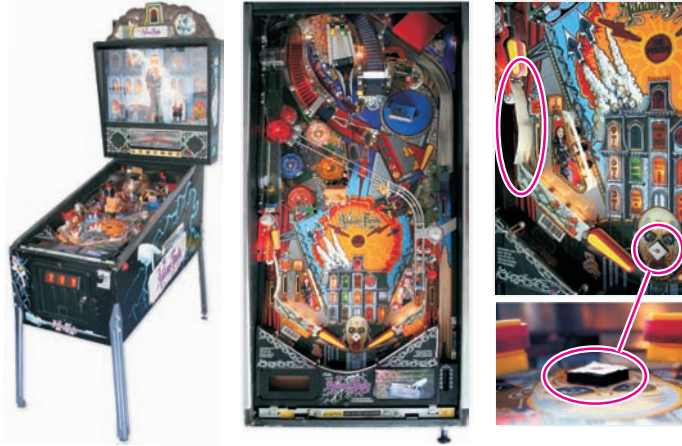

Figure 1: Left: pinball machine used for the present study. Middle: Close look at the build-in gadgets of the play field. Right: Zoom into the modified parts of the play field (side walls and central bump).

Compared to highly controlled and simplified lab settings, a pinball machine provides flow (according to the definition in [6]), a rich and complex feedback, acoustic and visual distractors, and a challenging behavioral task. These components are well-known ingredients for engaging and immersive game environments [7]. In case of the pinball machine model used in this study, this receives further evidence from the high sales figures that have made the Addams Family model the all-time popular pinball machine.

Given the reaction-time critical pinball game and the intrinsic delays imposed on the subjects by the BCI technology, it is very interesting to observe that subjects can manage to control and maintain the necessary timing and dynamics. The prediction of upcoming game situations and behavioral adaptation to the machine and BCI constraints are necessary ingredients to master this difficult task.

The following Sections Sec. 2 and Sec. 3 briefly introduce the used motor paradigm, spatial filter methods, the experimental paradigm, the decoding and machine learning techniques used, Sec. 4 provides the statistics and results, and finally a brief discussion is given in section Sec. 5.

## 2 Background

### 2.1 Neurophysiology

Macroscopic brain activity during resting wakefulness contains distinct rhythms located over various brain areas. Sensorimotor cortices show rhythmic macroscopic EEG oscillations ($\mu$-rhythm or sensorimotor rhythm, SMR), with spectral peak energies of about 8–14 Hz ($\alpha$-band) and/or 16–28 Hz ($\beta$-band) localized in the motor and somatosensory cortex ([8]).

A large class of EEG-based BCI systems relies on the fact that amplitude modulations of sensorimotor rhythms can be caused, e.g. by imagining movements. For example, the power of the $\mu$-rhythm decreases during imagined hand movements in the corresponding representation area which is located in the contralateral sensorimotor cortex. This phenomenon is called event-related desynchronization (ERD, [9, 10]), while the increase of band power is termed event-related synchronization (ERS). This may be observed, e.g., during motor imagery over flanking sensorimotor areas, possibly reflecting an 'surround inhibition' enhancing focal cortical activation, see [11, 10]. The exact location and the exact frequency band of the sensorimotor rhythm is subject-specific. Hence indi-

vidually optimized filters can increase the signal-to-noise ratio dramatically [12]. To this end, the CSP technique has proven to be useful.

## 2.2 Common Spatial Pattern (CSP) Analysis

Common Spatial Pattern and its extensions (e.g. [13, 14, 15, 16, 12]) is a technique to analyze multi-channel data based on recordings from two classes (conditions). It is used e.g. in BCI systems based on the modulation of brain rhythms. CSP yields a data-driven supervised decomposition of the signal parameterized by a matrix $W \in I\!R^{C \times C_0}$ ($C$ being the number of channels; $C_0 \leq C$) that projects the signal $\mathbf{x}(t) \in I\!R^C$ in the original sensor space to $\mathbf{x}_{\mathrm{CSP}}(t) \in I\!R^{C_0}$, which lives in the surrogate sensor space, as follows:

$$\mathbf{x}_{\mathrm{CSP}}(t) = \mathbf{W}^\top \mathbf{x}(t).$$

Each column vector of $\mathbf{W}$ represents a spatial filter. In particular CSP filters maximize the EEG signal's variance under one condition while simultaneously minimizing it for the other condition. Since variance of band-pass filtered signals is equal to band power, CSP analysis is applied to band-pass filtered signals in order to obtain an effective discrimination of mental states that are characterized by ERD/ERS effects (see above). In the example of left vs. right hand motor imagery, the CSP algorithm will find two groups of spatial filters. The first will show high band power during left hand motor imagery and low band power during right hand motor imagery, and the second vice versa.

Let $\mathbf{\Sigma}_i$ be the covariance matrix of the trial-concatenated matrix of dimension $[C \times T]$ (where $C$ is the number of electrodes and $T$ is the number of concatenated samples) belonging to the respective class $i \in \{1, 2\}$. The CSP analysis consists of calculating a matrix $\mathbf{W} \in I\!R^{C \times C}$ and a diagonal matrix $\mathbf{D}$ with elements in $[0, 1]$ such that

$$\mathbf{W}^\top \mathbf{\Sigma}_1 \mathbf{W} = \mathbf{D} \quad \text{and} \quad \mathbf{W}^\top \mathbf{\Sigma}_2 \mathbf{W} = \mathbf{I} - \mathbf{D} \qquad (1)$$

where $\mathbf{I} \in I\!R^{C \times C}$ is the identity matrix. This can be solved as a generalized eigenvalue problem. The projection that is given by the $i$-th column of matrix $\mathbf{W}$ has a relative variance of $d_i$ ($i$-th element of $\mathbf{D}$) for trials of class 1 and relative variance $1 - d_i$ for trials of class 2. If $d_i$ is near 1, the filter given by the $i$-th column of $W$ (i.e., the $i$th spatial filter) maximizes the variance for class 1, and since $1 - d_i$ is near 0, it also minimizes the variance for class 2. Typically one would retain projections corresponding to two or three of the highest eigenvalues $d_i$, i.e., CSP filters for class 1, and projections corresponding to the two or three lowest eigenvalues, i.e., CSP filters for class 2. For a detailed review of the CSP technique with respect to the application in BCI see [12].

# 3 Experiment

## 3.1 Paradigm

Standard EEG lab experiments typically realize an environment that avoids distractions in order to have maximum control over all parameters of the experiment. Since the subjects *respond* to a small number of artificial stimuli, a stimulus-locked averaging reveals the average characteristics of their brain response. If we are interested in understanding broader behavioral brain responses in cognitively demanding natural environments then stimulus/response-locked averaging may no longer be easily possible. The complexity in interaction may be caused by **(1)** a large number of possibilities to respond, **(2)** a large spread in response times and quality due to a rich environment (e.g. real objects that have a variety of physical properties), **(3)** a changing environment where the underlying nonstationarity is caused by a large number of states, and possibly by even more, but unknown influencing factors.

While the first steps towards complex paradigms use simulators that show an increased complexity but still allow complete introspection into the system state, it is evident that the interaction with real physical devices has an even higher complexity but also provides a rich multi-modal sensory experience for the user. However, gaining even only partial introspection into the system states of complex physical devices and into the interaction processes between the system and the mental processes of the user requires a huge effort.

Here modern machine learning and signal processing methods (e.g. [17, 18, 19, 20]) are helpful, since they have been developed to analyze EEG on a single trial basis (e.g. [21, 22]). They can adapt

to changing signal characteristics (e.g. [23, 24, 25]) and they can deal with missing and noisy data [26, 27] – even beyond the field of computational neuroscience and BCI [28].

## 3.2 Setup

In this study seven subjects played with the pinball machine. They were known for well-classifiable EEG signals in simple BCI applications. One subject played successfully and enjoyed it, but was excluded from further analysis as his/her games had not been video-taped. From the remaining six subjects, three managed to acquire good control, played very successfully and enjoyed this experience. One subject managed to get limited control and reported to enjoy the games although some of his/her scores were close to chance. The performance of these four subjects was measured in a rigorous manner. The remaining two subjects could not establish reliable control and were also excluded from further analysis.

An overview of the technical setup and the data processing steps involved is given by Fig. 2. The experiment was organized in several stages: the calibration of the BCI system (Sec. 3.3), the fine-tuning of parameters in a simple cursor feedback paradigm (Sec. 3.4), the application of the BCI control system during pinball games (Sec. 3.5), the pseudo-random control of pinball games (Sec. 3.6), and ball insertions without any paddle activity (Sec. 3.7).

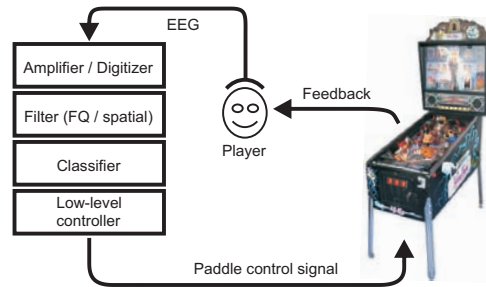

Figure 2: Schematic view of the BCI-controlled pinball machine. The user's EEG signals upon motor imagery are amplified, digitized, filtered in the frequency domain and the spatial domain by CSP. Band power features are extracted and classified. The classifier output is translated by a low-level controller into paddle movements.

## 3.3 Calibration of the BCI system

The BCI system was calibrated individually for each of the subjects (*VPMa*, *VPks*, *VPzq*, *VPlf*) to discriminate two classes of motor imagery (left hand and right hand). The calibration procedure followed a standard Berlin BCI (BBCI) paradigm based on spatial filters and oscillatory features that avoids and prevents the use of class-correlated EOG or EMG artefacts (see [29, 28] for details). Visualizing the spatial filters and the resulting patterns of activity showed that EOG or EMG components were disregarded for the calibration of the BCI system. For the calibration, 100 (*VPMa*) or 75 (*VPks*, *VPzq*, *VPlf*) trials of motor imagery were collected for each class. For every trial of 4–5s duration, the class of the motor imagery was indicated on a computer screen by visual cues. The calibration procedure included the determination of a subject-specific frequency band for the mu-rhythm (see Sec. 2.1), filtering the 64-channel EEG-data to this band, the determination of class-discriminant spatial filters with Common Spatial Pattern (CSP, see Sec. 2.2), and the training of a regularized linear classification method (LDA) based on the power features of the filtered data. All subjects showed a crossvalidation error below 10% on the calibration data.

## 3.4 Cursor feedback control by BCI

The bias of the classifier, a gain factor and thresholds for an idle-class (for classifier outputs close to the decision plane) were adapted during a short control task running on a computer screen. The subject had to control a horizontally moving cursor to a target on the left or right side of the screen

for approximately 2 minutes while fixating a cross in the center. During this procedure the above mentioned parameters were fine-tuned according to the test persons's ratings. The goals were to determine parameter values that translate the classifier output to a suitable range for the final application and – for the test persons – to reach a subjective feeling of control. For an exhaustive study on the role of bias adaptation in BCI, especially in the context of changing from calibration to feedback, see [30, 24].

## 3.5 Pinball control by BCI

A real, physical pinball machine (in our study an Addams Family model) needs good control in terms of classification accuracy and timing (dynamics).

The subject has to learn the physical properties of the machine to play well. The subject's expectation needs to be trained as bumpers, magnets like "The Power" and many other built-in sources of surprise (see middle image in Fig. 1) can cause the ball to go into rather unpredictable directions. This interaction with the pinball machine makes the game interesting and challenging. Fast brain dynamics that participate in the eye-hand coordination and visual memory play an essential role to cope with these difficulties. The task difficulty increases further, as with any game, there is a strong emotional engagement of the subject which gives rise to non-stationarities in the statistics. Moreover the physical machine is very noisy and distracting due to its various sources of visual and auditory stimulation, and only a small percentage of these stimulations is task relevant.

Three modifications were implemented in order to reduce the frequency of manual ball launches (1 and 3) and to increase the frequency of balls passing the paddle areas (1 and 2). While the original character of the game was not changed, the modifications introduced slight simplification to conduct the experiment. The right image of Fig. 1 depicts the modifications:

1. side limits that prevents balls from exiting without passing the paddles

2. a soft central bump in front of the paddles that biases balls to pass one of the paddles rather than exiting in a perfect vertical trajectory. This is necessary, as the classifier output could not activate both paddles at exactly the same time.

3. a reduced slope of the game field (about half the original slope), that somewhat slows down the game speed.

During the BCI-controlled gaming ("**bci**" control mode), the subject sat in front of the pinball machine, hands resting on the arm rests except for short times when new balls had to be launched with the pulling lever. The EEG signals recorded in the previous 500ms were translated by the BCI system into a control signal. A simple low-level control mechanism was implemented in software that translated the continuous classifier output by thresholding into a three-class signal (left flipper, idle, right flipper) using the thresholds pre-determined during the cursor control (see Sec. 3.4). Furthermore it introduced a logic that translated a very long lasting control signal for the left or right class into a hold-and-shoot mechanism. This allowed the user to catch slow balls rolling sideways down towards a paddle. The user played several games of 10 to 12 balls each. Performance was observed in terms of the playing time per ball, the score per game and the number of high-quality shots. The latter were defined by the presence of one of the following two conditions, which have been evaluated in an offline video analysis of the game: **(1)** a precisely timed shot that hit the ball by the center of the paddle and drives it into one of the scoring zones of the lower half of the field and **(2)** a precisely timed shot that drives the ball directly into the upper half of the field.

## 3.6 Pseudo random control mode

This "**rand**" control mode was incorporated into the experimental setup in order to deliver a fair performance baseline. Here, the BCI system was up and running with the same settings as in the BCI-controlled pinball game, but no player was present. Instead an EEG file previously recorded during the BCI-controlled pinball game was fed into the BCI system and generated the control signal for the pinball machine. These signals produced the same statistics of paddle movements as in the real feedback setting. But as the balls were launched at random time points, the paddle behavior was not synchronized with the ball positions. Therefore, the pseudo random control mode marks

the chance level of the system. In this mode several games of 10-12 balls each were performed. The same performance measures were applied as for BCI-controlled gaming.

### 3.7 No control mode

For performance comparisons, two performance ratings (time per ball and points per game) were also taken for a series of balls that were launched without any paddle movements ("**none**" control mode).

## 4 Results

As video recordings have been available for the four subjects, a detailed analysis of the game performances was possible. It is introduced for the example of the best subject *VPMa* in Fig. 3. The analysis compares three different scoring measures for BCI control (**bbci**), pseudo-random control (**rand**) and no control (**none**) and shows the histogram of high-quality shots per ball. The average

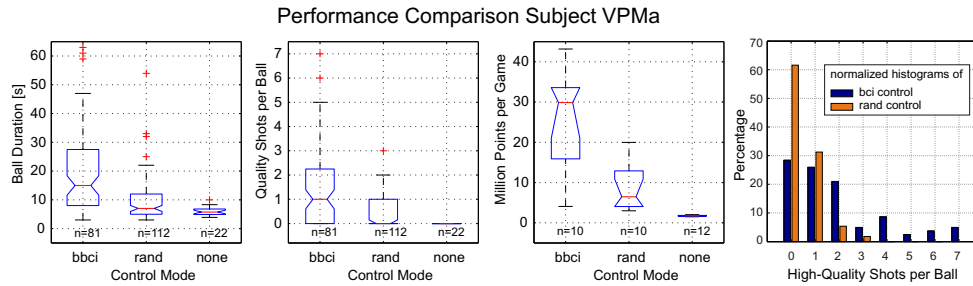

Figure 3: Performance comparison for three control modes of the pinball machine and the normalized histograms of high-quality shots per ball for subject *VPMa*.

ball duration (median) is significantly higher for the BCI-controlled gaming (average of 15s over 81 balls) than for the pseudo-random control (average of 8s over 112 balls). A confidence interval is reflected by the notches above and below the median values in the boxplot of Fig. 3. Boxes whose notches do not overlap indicate that the medians of the two groups differ at the 5% significance level. The increased average ball duration under BCI control is caused by the larger number of high-quality shots per ball. While in pseudo-random control only 7% of the balls scored more than one high-quality shot per ball, this rate raises drastically to 45% for the BCI control of subject *VPMa*. A comparison of the game scores for 10 games of BCI control and 10 games of pseudo-random control shows, that these differ even stronger due to the nonlinear characteristic of the score. The rightmost plot in Fig. 3 shows the normalized histograms of the high-quality shots.

The pooled data of all four subjects in Fig. 4 reflects these performance differences to a large extend. Again, BCI control is significantly superior to the pseudo random control. The difference in normalized histograms between BCI control and pseudo random control reveals, that even for the pooled data BCI-controlled games more often have a larger number of high-quality shots.

Not surprisingly, the BCI-controlled games showed a number of paddle movements in moments, when no ball was in the vicinity of the paddles. These so-called false hits are indirectly reflected in the performance measures for the pseudo-random control. As pseudo-random control mode was able to gain significantly better results than no control at all (see e.g. modes **rand** and **none** in Fig. 3), these false hits can not be neglected. In order to study this issue, the pseudo-random control was based on an EEG file, which had been previously recorded during the BCI-controlled gaming, the dynamics of the paddle movements was identical during both of these control modes. Under these very similar conditions, the higher scores of the BCI control must be credited to the control ability of the BCI user, especially to the precise timing of a large number of paddle shots.

A video of the gaming performance which provides an impression of the astonishing level of timing and dynamical control – much better than the figures can show – is available under `http://www.bbci.de/supplementary/`. It should be added that for this experiment it was very easy to recruit highly motivated subjects, who enjoyed the session.

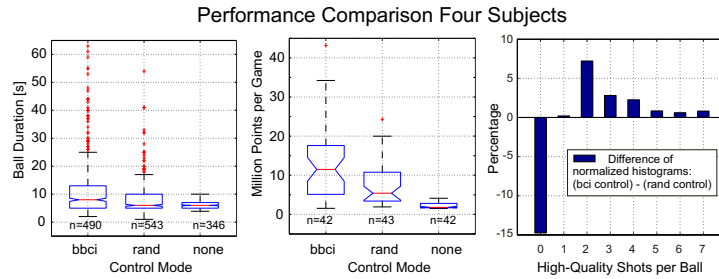

Figure 4: Performance comparison for combined data of four subjects (*VPMa*, *VPks*, *VPzq*, *VPlf* ).

## 5    Discussion

To date, BCI is mainly perceived as an opportunity for the disabled to regain interaction with their environment, say, through BCI actuated spelling or other forms of BCI control.

The present study is relevant to rehabilitation since it explores the limits of BCI with respect to timing, dynamics and speed of interaction in a difficult real-time task. We would, however, like to re-iterate to consider machine learning methods developed in BCI also as novel powerful tools for the neurosciences – not only when operated invasively for harvesting on local field potentials (LFP) and on micro electrode array data [1, 2, 3] or for decoding functional MRI [31] – but also for non-invasive, low-risk EEG-BCI.

An important novel aspect of our study was to analyze EEG recorded during predictive behavior, in other words we made use of the subject's expectation and experience of the system delay. Learning curves and traces of adaptation on the subject side, the use of error potentials as well as emerging subject specific strategy differences and many other exciting question must remain untouched in this first study. Emotion, surprise and other mental states or cognitive processes that play an important role in such complex real-time paradigms still await their consideration in future studies.

### Acknowledgments

We thank Brain Products GmbH for funding and for help with the preparation of the pinball machine. Funding by the *European Community* under the PASCAL Network of Excellence (IST-2002-506778) and under the FP7 Programme (TOBI ICT-2007-224631), by the *Bundesministerium für Bildung und Forschung (BMBF)* (FKZ 01IBE01A and FKZ 16SV2231) and by the *Deutsche Forschungsgemeinschaft (DFG)* (VitalBCI MU 987/3-1) is gratefully acknowledged. Last but not least, we would like to thank our reviewers for their valuable comments.

## References

[1] J. M. Carmena, M. A. Lebedev, R. E. Crist, J. E. O'Doherty, D. M. Santucci, D. F. Dimitrov, P. G. Patil, C. S. Henriquez, and M. A. Nicolelis. Learning to control a brain-machine interface for reaching and grasping by primates. *PLoS Biol*, E42, 2003.

[2] D. M. Taylor, S. I. Tillery, and A. B. Schwartz. Direct cortical control of 3D neuroprosthetic devices. *Science*, 296:1829–1832, 2002.

[3] L.R. Hochberg, M.D. Serruya, G.M. Friehs, J.A. Mukand, M. Saleh, A.H. Caplan, A. Branner, D. Chen, R.D. Penn, and J.P. Donoghue. Neuronal ensemble control of prosthetic devices by a human with tetraplegia. *Nature*, 442(7099):164–171, July 2006.

[4] J. R. Wolpaw and D. J. McFarland. Control of a two-dimensional movement signal by a noninvasive brain-computer interface in humans. *Proc Natl Acad Sci USA*, 101(51):17849–17854, 2004.

[5] Andrea Kübler and Klaus-Robert Müller. An introduction to brain computer interfacing. In Guido Dornhege et al., editors, *Toward Brain-Computer Interfacing*, pages 1–25. MIT press, Cambridge, MA, 2007.

[6] W. A. IJsselsteijn, H. H. Nap, Y. A. W. de Kort, K. Poels andA. Jurgelionis, and F. Bellotti. Characterizing and measuring user experiences in digital games. In *Proceedings of the ACE*, Salzburg, 2007.

[7] C. Jennett, A. L. Cox, P. Cairns, S. Dhoparee, A. Epps, T. Tijs, and A. Walton. Measuring and defining the experience of immersion in games. *International Journal of Human Computer Studies*, 2008.

[8] H. Jasper and H.L. Andrews. Normal differentiation of occipital and precentral regions in man. *Arch. Neurol. Psychiat. (Chicago)*, 39:96–115, 1938.

[9] Gert Pfurtscheller and F.H. Lopes da Silva. Event-related EEG/MEG synchronization and desynchronization: basic principles. *Clin Neurophysiol*, 110(11):1842–1857, Nov 1999.

[10] G. Pfurtscheller, C. Brunner, A. Schlögl, and F.H. Lopes da Silva. Mu rhythm (de)synchronization and EEG single-trial classification of different motor imagery tasks. *NeuroImage*, 31(1):153–159, 2006.

[11] C. Neuper and G. Pfurtscheller. Evidence for distinct beta resonance frequencies in human EEG related to specific sensorimotor cortical areas. *Clin Neurophysiol*, 112:2084–2097, 2001.

[12] Benjamin Blankertz, Ryota Tomioka, Steven Lemm, Motoaki Kawanabe, and Klaus-Robert Müller. Optimizing spatial filters for robust EEG single-trial analysis. *IEEE Signal Proc Magazine*, 25(1):41–56, January 2008.

[13] Keinosuke Fukunaga. *Introduction to statistical pattern recognition*. Academic Press, Boston, 2nd edition edition, 1990.

[14] Z. J. Koles. The quantitative extraction and topographic mapping of the abnormal components in the clinical EEG. *Electroencephalogr Clin Neurophysiol*, 79(6):440–447, 1991.

[15] Steven Lemm, Benjamin Blankertz, Gabriel Curio, and Klaus-Robert Müller. Spatio-spectral filters for improving classification of single trial EEG. *IEEE Trans Biomed Eng*, 52(9):1541–1548, 2005.

[16] Guido Dornhege, Benjamin Blankertz, Matthias Krauledat, Florian Losch, Gabriel Curio, and Klaus-Robert Müller. Optimizing spatio-temporal filters for improving brain-computer interfacing. In *Advances in Neural Inf. Proc. Systems (NIPS 05)*, volume 18, pages 315–322, Cambridge, MA, 2006. MIT Press.

[17] B. Schölkopf and A.J. Smola. *Learning with Kernels*. MIT Press, Cambridge, MA, 2002.

[18] K.-R. Müller, S. Mika, G. Rätsch, K. Tsuda, and B. Schölkopf. An introduction to kernel-based learning algorithms. *IEEE Neural Networks*, 12(2):181–201, May 2001.

[19] Klaus-Robert Müller, Charles W. Anderson, and Gary E. Birch. Linear and non-linear methods for brain-computer interfaces. *IEEE Trans Neural Sys Rehab Eng*, 11(2):165–169, 2003.

[20] S. Haykin. *Neural Networks : A Comprehensive Foundation*. Macmillan, New York, 1994.

[21] N.J. Hill, T. N. Lal, M. Tangermann, T. Hinterberger, G. Widman, C. E. Elger, B. Schölkopf, and N. Birbaumer. Classifying event-related desynchronization in EEG, ECoG and MEG signals. In Guido Dornhege et al., editors, *Toward Brain-Computer Interfacing*, pages 235–260. MIT press, Cambridge, MA, 2007.

[22] Benjamin Blankertz, Florian Losch, Matthias Krauledat, Guido Dornhege, Gabriel Curio, and Klaus-Robert Müller. The Berlin Brain-Computer Interface: Accurate performance from first-session in BCI-naive subjects. *IEEE Trans Biomed Eng*, 2008. in press.

[23] Matthias Krauledat, Michael Schröder, Benjamin Blankertz, and Klaus-Robert Müller. Reducing calibration time for brain-computer interfaces: A clustering approach. In B. Schölkopf, J. Platt, and T. Hoffman, editors, *Advances in Neural Information Processing Systems 19*, pages 753–760, Cambridge, MA, 2007. MIT Press.

[24] Masashi Sugiyama, Matthias Krauledat, and Klaus-Robert Müller. Covariate shift adaptation by importance weighted cross validation. *Journal of Machine Learning Research*, 8:1027–1061, 2007.

[25] Pradeep Shenoy, Matthias Krauledat, Benjamin Blankertz, Rajesh P. N. Rao, and Klaus-Robert Müller. Towards adaptive classification for BCI. *J Neural Eng*, 3(1):R13–R23, 2006.

[26] Guido Dornhege, Matthias Krauledat, Klaus-Robert Müller, and Benjamin Blankertz. General signal processing and machine learning tools for BCI. In Guido Dornhege et al., editors, *Toward Brain-Computer Interfacing*, pages 207–233. MIT Press, Cambridge, MA, 2007.

[27] Matthias Krauledat, Guido Dornhege, Benjamin Blankertz, and Klaus-Robert Müller. Robustifying EEG data analysis by removing outliers. *Chaos and Complexity Letters*, 2(3):259–274, 2007.

[28] Klaus-Robert Müller, Michael Tangermann, Guido Dornhege, Matthias Krauledat, Gabriel Curio, and Benjamin Blankertz. Machine learning for real-time single-trial EEG-analysis: From brain-computer interfacing to mental state monitoring. *J Neurosci Methods*, 167(1):82–90, 2008.

[29] Benjamin Blankertz, Guido Dornhege, Matthias Krauledat, Klaus-Robert Müller, and Gabriel Curio. The non-invasive Berlin Brain-Computer Interface: Fast acquisition of effective performance in untrained subjects. *NeuroImage*, 37(2):539–550, 2007.

[30] Matthias Krauledat, Pradeep Shenoy, Benjamin Blankertz, Rajesh P. N. Rao, and Klaus-Robert Müller. Adaptation in CSP-based BCI systems. In Guido Dornhege et al., editors, *Toward Brain-Computer Interfacing*, pages 305–309. MIT Press, Cambridge, MA, 2007.

[31] J.D. Haynes and G. Rees. Decoding mental states from brain activity in humans. *Nature Reviews Neuroscience*, 7:523–534, 2006.

